# Learning to Segment Images Using Dynamic Feature Binding

**Michael C. Moser**
Dept. of Comp. Science &
Inst. of Cognitive Science
University of Colorado
Boulder, CO 80309–0430

**Richard S. Zemel**
Dept. of Comp. Science
University of Toronto
Toronto, Ontario
Canada  M5S 1A4

**Marlene Behrmann**
Dept. of Psychology &
Faculty of Medicine
University of Toronto
Toronto, Ontario
Canada  M5S 1A1

## Abstract

Despite the fact that complex visual scenes contain multiple, overlapping objects, people perform object recognition with ease and accuracy. One operation that facilitates recognition is an early segmentation process in which features of objects are grouped and labeled according to which object they belong. Current computational systems that perform this operation are based on predefined grouping heuristics. We describe a system called MAGIC that *learns* how to group features based on a set of presegmented examples. In many cases, MAGIC discovers grouping heuristics similar to those previously proposed, but it also has the capability of finding nonintuitive structural regularities in images. Grouping is performed by a relaxation network that attempts to dynamically bind related features. Features transmit a complex-valued signal (amplitude and phase) to one another; binding can thus be represented by phase locking related features. MAGIC's training procedure is a generalization of recurrent back propagation to complex-valued units.

When a visual image contains multiple, overlapping objects, recognition is difficult because features in the image are not grouped according to which object they belong. Without the capability to form such groupings, it would be necessary to undergo a massive search through all subsets of image features. For this reason, most machine vision recognition systems include a component that performs feature grouping or *image segmentation* (e.g., Guzman, 1968; Lowe, 1985; Marr, 1982).

A multitude of heuristics have been proposed for segmenting images. Gestalt psychologists have explored how people group elements of a display and have suggested a range of grouping principles that govern human perception (Rock & Palmer, 1990). Computer vision researchers have studied the problem from a more computational perspective. They have investigated methods of grouping elements of an image based on *nonaccidental regularities*—feature combinations that are unlikely to occur by chance when several objects are juxtaposed, and are thus indicative of a single object (Kanade, 1981; Lowe & Binford, 1982).

In these earlier approaches, the researchers have hypothesized a set of grouping heuristics and then tested their psychological validity or computational utility. In our work, we have taken an *adaptive* approach to the problem of image segmentation in which a system learns how to group features based on a set of examples. We call the system MAGIC, an acronym for multiple-object adaptive grouping of image components. In many cases MAGIC discovers grouping heuristics similar to those proposed in earlier work, but it also has the capability of finding nonintuitive structural regularities in images.

MAGIC is trained on a set of presegmented images containing multiple objects. By "presegmented," we mean that each image feature is labeled as to which object it belongs. MAGIC learns to detect configurations of the image features that have a consistent labeling in relation to one another across the training examples. Identifying these configurations allows MAGIC to then label features in novel, unsegmented images in a manner consistent with the training examples.

## 1  REPRESENTING FEATURE LABELINGS

Before describing MAGIC, we must first discuss a representation that allows for the labeling of features. Von der Malsburg (1981), von der Malsburg & Schneider (1986), Gray et al. (1989), and Eckhorn et al. (1988), among others, have suggested a biologically plausible mechanism of labeling through temporal correlations among neural signals, either the relative timing of neuronal spikes or the synchronization of oscillatory activities in the nervous system. The key idea here is that each processing unit conveys not just an activation value—average firing frequency in neural terms—but also a second, independent value which represents the relative *phase* of firing. The dynamic grouping or *binding* of a set of features is accomplished by aligning the phases of the features. Recent work (Goebel, 1991; Hummel & Biederman, in press) has used this notion of dynamic binding for grouping image features, but has been based on relatively simple, predetermined grouping heuristics.

## 2  THE DOMAIN

Our initial work has been conducted in the domain of two-dimensional geometric contours, including rectangles, diamonds, crosses, triangles, hexagons, and octagons. The contours are constructed from four primitive feature types—oriented line segments at 0°, 45°, 90°, and 135°—and are laid out on a 15 × 20 grid. At each location on the grid are units, called *feature units*, that detect each of the four primitive feature types. In our present experiments, images contain two contours. Contours are not permitted to overlap in their activation of the same feature unit.

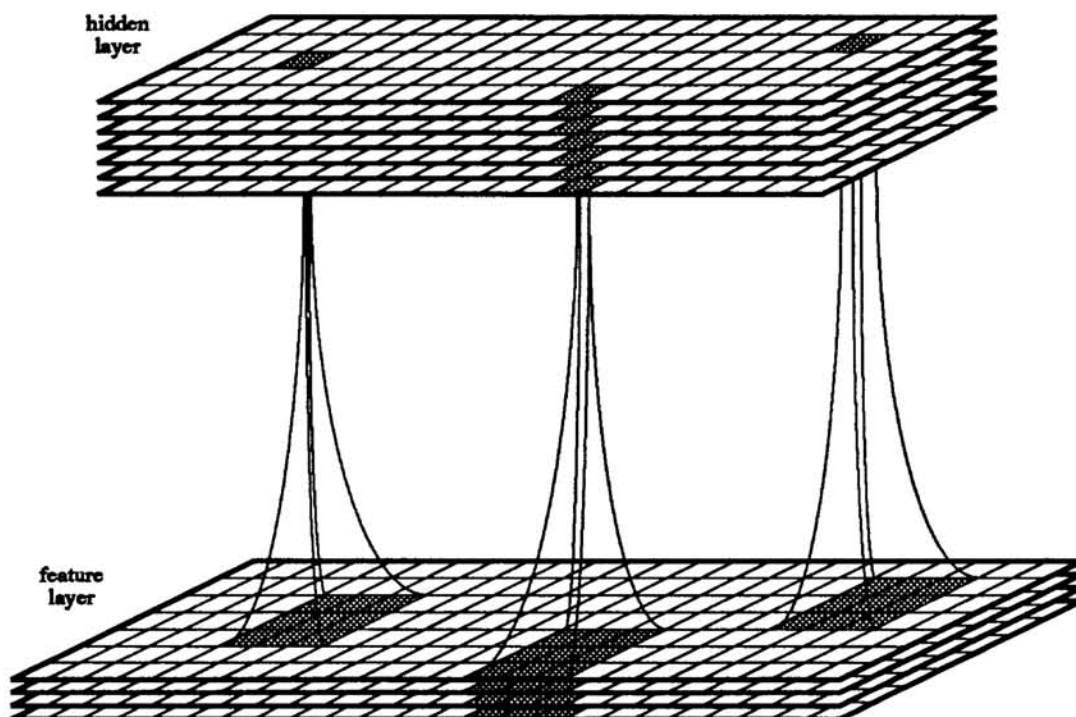

Figure 1: The architecture of MAGIC. The lower layer contains the feature units; the upper layer contains the hidden units. Each layer is arranged in a spatiotopic array with a number of different feature types at each position in the array. Each plane in the feature layer corresponds to a different feature type. The grayed hidden units are reciprocally connected to all features in the corresponding grayed region of the feature layer. The lines between layers represent projections in both directions.

## 3   THE ARCHITECTURE

The input to MAGIC is a pattern of activity over the feature units indicating which features are present in an image. The initial phases of the units are random. MAGIC's task is to assign appropriate phase values to the units. Thus, the network performs a type of pattern completion.

The network architecture consists of two layers of units, as shown in Figure 1. The lower (input) layer contains the feature units, arranged in spatiotopic arrays with one array per feature type. The upper layer contains hidden units that help to align the phases of the feature units; their response properties are determined by training. Each hidden unit is reciprocally connected to the units in a local spatial region of all feature arrays. We refer to this region as a *patch*; in our current simulations, the patch has dimensions 4 × 4. For each patch there is a corresponding fixed-size *pool* of hidden units. To achieve uniformity of response across the image, the pools are arranged in a spatiotopic array in which neighboring pools respond to neighboring patches and the weights of all pools are constrained to be the same.

The feature units activate the hidden units, which in turn feed back to the feature units. Through a relaxation process, the system settles on an assignment of phases to the features.

## 4   NETWORK DYNAMICS

Formally, the response of each feature unit $i$, $x_i$, is a complex value in polar form, $(a_i, p_i)$, where $a_i$ is the amplitude or activation and $p_i$ is the phase. Similarly, the response of each hidden unit $j$, $y_j$, has components $(b_j, q_j)$. The weight connecting unit $i$ to unit $j$, $w_{ji}$, is also complex valued, having components $(\rho_{ji}, \theta_{ji})$. The activation rule we propose is a generalization of the dot product to the complex domain:

$$
\begin{aligned}
net_j &= \mathbf{x} \cdot \mathbf{w}_j \\
&= \sum_i x_i \bar{w}_{ji} \\
&= \left( \left[ \left( \sum_i a_i \rho_{ji} \cos(p_i - \theta_{ji}) \right)^2 + \left( \sum_i a_i \rho_{ji} \sin(p_i - \theta_{ji}) \right)^2 \right]^{\frac{1}{2}}, \right. \\
&\qquad \left. \tan^{-1} \left[ \frac{\sum_i a_i \rho_{ji} \sin(p_i - \theta_{ji})}{\sum_i a_i \rho_{ji} \cos(p_i - \theta_{ji})} \right] \right)
\end{aligned}
$$

where $net_j$ is the net input to hidden unit $j$. The net input is passed through a squashing nonlinearity that maps the amplitude of the response from the range $0 \rightarrow \infty$ to $0 \rightarrow 1$ but leaves the phase unaffected:

$$
y_j = \frac{net_j}{|net_j|} \left( 1 - e^{-|net_j|^2} \right).
$$

The flow of activation from the hidden layer to the feature layer follows the same dynamics, although in the current implementation the amplitudes of the features are clamped, hence the top-down flow affects only the phases. One could imagine a more general architecture in which the relaxation process determined not only the phase values, but cleaned up noise in the feature amplitudes as well.

The intuition underlying the activation rule is as follows. The activity of a hidden unit, $b_j$, should be monotonically related to how well the feature response pattern matches the hidden unit weight vector, just as in the standard real-valued activation rule. Indeed, one can readily see that if the feature and weight phases are equal ($p_i = \theta_{ji}$), the rule for $b_j$ reduces to the real-valued case. Even if the feature and weight phases differ by a constant ($p_i = \theta_{ji} + c$), $b_j$ is unaffected. This is a critical property of the activation rule: Because *absolute* phase values have no intrinsic meaning, the response of a unit should depend only on the *relative* phases. The activation rule achieves this by essentially ignoring the average difference in phase between the feature units and the weights. The hidden phase, $q_j$, reflects this average difference.

## 5   LEARNING ALGORITHM

During training, we would like the hidden units to learn to detect configurations of features that reliably indicate phase relationships among the features. We have experimented with a variety of training algorithms. The one with which we have had greatest success involves running the network for a fixed number of iterations and, after each iteration, attempting to adjust the weights so that the feature phase pattern will match a target phase pattern. Each training trial proceeds as follows:

1. A training example is generated at random. This involves selecting two contours and instantiating them in an image. The features of one contour have *target* phase 0° and the features of the other contour have target phase 180°.

2. The training example is presented to MAGIC by clamping the amplitude of a feature unit to 1.0 if its corresponding image feature is present, or 0.0 otherwise. The phases of the feature units are set to random values in the range 0° to 360°.

3. Activity is allowed to flow from the feature units to the hidden units and back to the feature units. Because the feature amplitudes are clamped, they are unaffected.

4. The new phase pattern over the feature units is compared to the target phase pattern (see step 1), and an error measure is computed:

$$E = -\left(\sum_i a_i \cos(\bar{p}_i - p_i)\right)^2 - \left(\sum_i a_i \sin(\bar{p}_i - p_i)\right)^2,$$

where $\bar{p}$ is the target phase pattern. This error ignores the absolute difference between the target and actual phases. That is, $E$ is minimized when $\bar{p}_i - p_i$ is a constant for all $i$, regardless of the value of $\bar{p}_i - p_i$.

5. Using a generalization of back propagation to complex valued units, error gradients are computed for the feature-to-hidden and hidden-to-feature weights.

6. Steps 3–5 are repeated for a maximum of 30 iterations. The trial is terminated if the error increases on five consecutive iterations.

7. Weights are updated by an amount proportional to the average error gradient over iterations.

Learning is more robust when the feature-to-hidden weights are constrained to be symmetric with the hidden-to-feature weights. For complex weights, symmetry means that the weight from feature unit $i$ to hidden unit $j$ is the complex conjugate of the weight from hidden unit $j$ to feature unit $i$. Weight symmetry ensures that MAGIC will converge to a fixed point. (The proof is based on discrete-time update and a two-layer architecture with sequential layer updates and no intralayer connections.)

Simulations reported below use a learning rate of .005 for the amplitudes and 0.02 for the phases. About 10,000 learning trials are required for stable performance, although MAGIC rapidly picks up on the most salient aspects of the domain.

## 6   SIMULATION RESULTS

We trained a network with 20 hidden units per pool on images containing either two rectangles, two diamonds, or a rectangle and a diamond. The shapes were of varying size and appeared in various locations. A subset of the resulting weights are shown in Figure 2. Each hidden unit attempts to detect and reinstantiate activity patterns that match its weights. One clear and prevalent pattern in the weights is the collinear arrangement of segments of a given orientation, all having the same phase value. When a hidden unit having weights of this form responds to a patch of the feature array, it tries align the phases of the patch with the phases of its weight vector. By synchronizing the phases of features, it acts to group the features. Thus, one can interpret the weight vectors as the rules by which features are grouped.

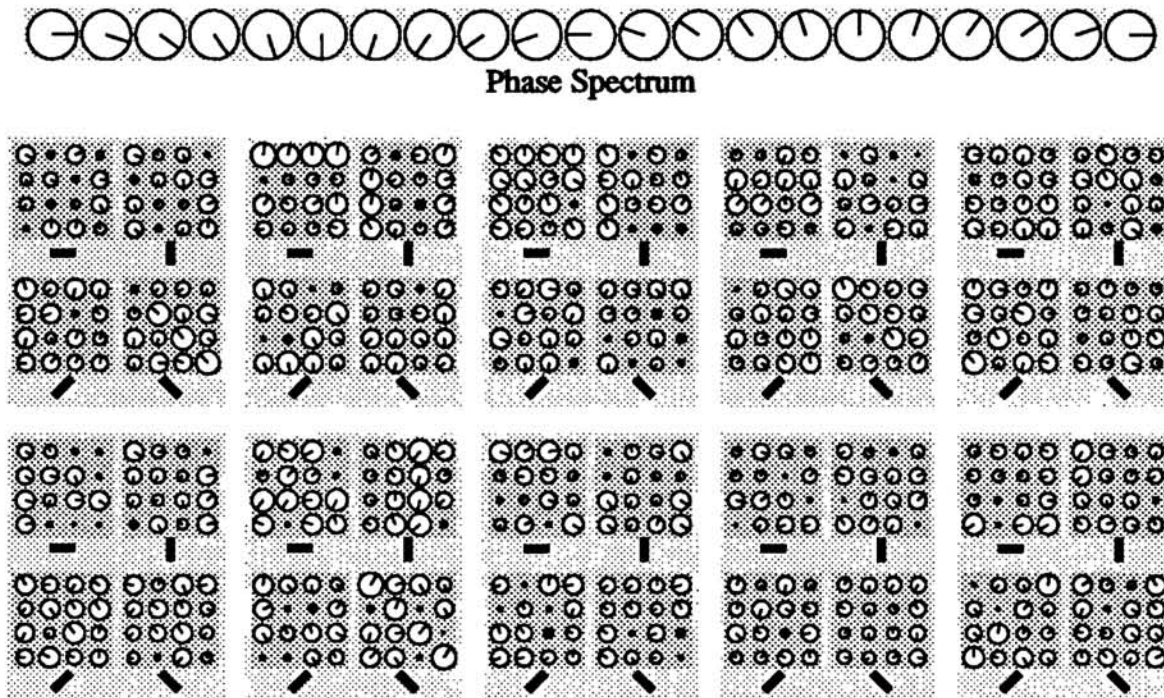

Phase Spectrum

Figure 2: Sample of feature-to-hidden weights learned by MAGIC. The area of a circle represents the amplitude of a weight, the orientation of the internal tick mark represents the phase angle. The weights are arranged such that the connections into each hidden unit are presented on a light gray background. Each hidden unit has a total of 64 incoming weights—4 × 4 locations in its receptive field and four feature types at each location. The weights are further grouped by feature type, and for each feature type they are arranged in a 4 × 4 pattern homologous to the image patch itself.

Whereas traditional grouping principles indicate the conditions under which features should be bound together as part of the same object, the grouping principles learned by MAGIC also indicate when features should be segregated into different objects. For example, the weights of the vertical and horizontal segments are generally 180° out of phase with the diagonal segments. This allows MAGIC to segregate the vertical and horizontal features of a rectangle from the diagonal features of a diamond. We had anticipated that the weights to each hidden unit would contain two phase values at most because each image patch contains at most two objects. However, some units make use of three or more phases, suggesting that the hidden unit is performing several distinct functions. As is the usual case with hidden unit weights, these patterns are difficult to interpret.

Figure 3 presents an example of the network segmenting an image. The image contains two diamonds. The top left panel shows the features of the diamonds and their initial random phases. The succeeding panels show the network's response during the relaxation process. The lower right panel shows the network response at equilibrium. Features of each object have been assigned a uniform phase, and the two objects are 180° out of phase. The task here may appear simple, but it is quite challenging due to the illusory diamond generated by the overlapping diamonds.

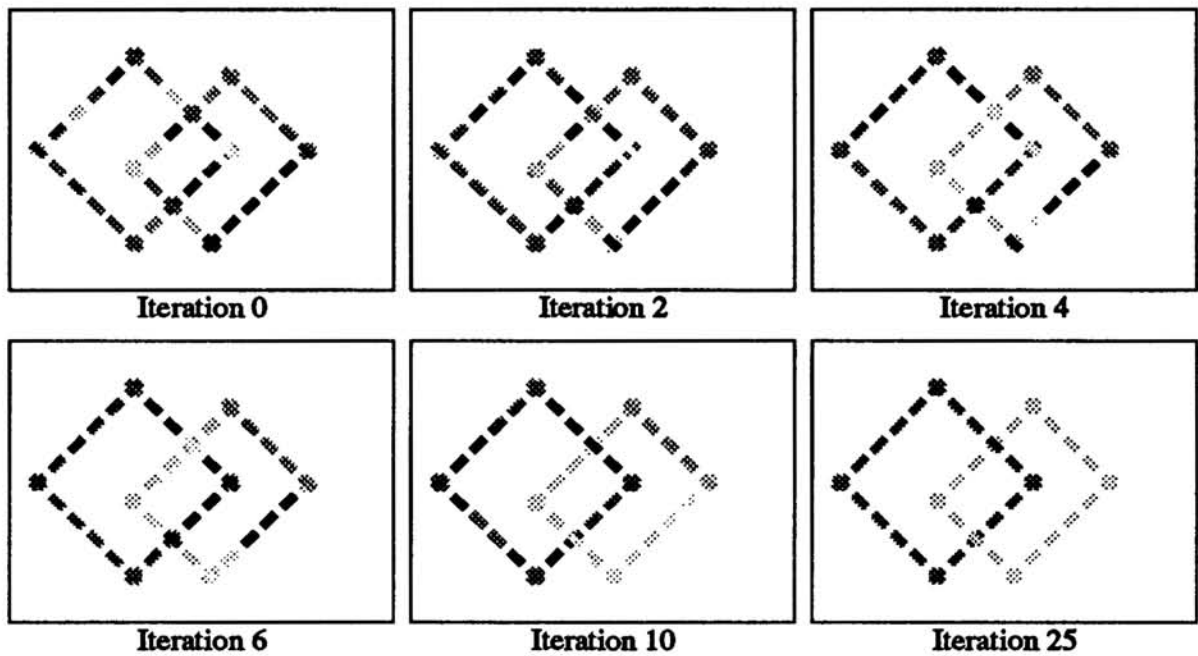

Figure 3: An example of MAGIC segmenting an image. The "iteration" refers to the number of times activity has flowed from the feature units to the hidden units and back. The phase value of a feature is represented by a gray level. The periodic phase continuum can only be approximated by the linear gray level continuum, but the basic information is conveyed nonetheless.

## 7    CURRENT DIRECTIONS

We are currently extending MAGIC in several directions, which we outline here.

- A natural principle for the hierarchical decomposition of objects emerges from the relative frequency of feature configurations during training. More frequent configurations result in a robust hidden representation, and hence the features forming these configurations will be tightly coupled. A coarse quantization of phases will lead to parses of the image in which only the highest frequency configurations are considered as "objects." Finer quantizations will lead to a further decomposition of the image. Thus, the continuous phase representation allows for the construction of hierarchical descriptions of objects.

- Spatially local grouping principles are unlikely to be sufficient for the image segmentation task. Indeed, we have encountered incorrect solutions produced by MAGIC that are locally consistent but globally inconsistent. To solve this problem, we are investigating an architecture in which the image is processed at several spatial scales simultaneously.

- Simulations are also underway to examine MAGIC's performance on real-world images—overlapping handwritten letters and digits—where it is somewhat less clear to which types of patterns the hidden units should respond.

- Zemel, Williams, and Mozer (to appear) have proposed a mathematical framework that—with slight modifications to the model—allow it to be interpreted

as a mean-field approximation to a stochastic phase model.

- Behrmann, Zemel, and Mozer (to appear) are conducting psychological experiments to examine whether limitations of the model match human limitations.

## Acknowledgements

This research was supported by NSF Presidential Young Investigator award IRI–9058450, grant 90–21 from the James S. McDonnell Foundation, and DEC external research grant 1250 to MM, and by a National Sciences and Engineering Research Council Postgraduate Scholarship to RZ. Our thanks to Paul Smolensky, Chris Williams, Geoffrey Hinton, and Jürgen Schmidhuber for helpful comments regarding this work.

## References

Eckhorn, R., Bauer, R., Jordan, W., Brosch, M., Kruse, W., Munk, M., & Reitboek, H. J. (1988). Coherent oscillations: A mechanism of feature linking in the visual cortex? *Biological Cybernetics*, *60*, 121–130.

Goebel, R. (1991). An oscillatory neural network model of visual attention, pattern recognition, and response generation. Manuscript in preparation.

Gray, C. M., Koenig, P., Engel, A. K., & Singer, W. (1989). Oscillatory responses in cat visual cortex exhibit intercolumnar synchronization which reflects global stimulus properties. *Nature (London)*, *338*, 334–337.

Guzman, A. (1968). Decomposition of a visual scene into three-dimensional bodies. *AFIPS Fall Joint Computer Conference*, *33*, 291–304.

Hummel, J. E., & Biederman, I. (1992). Dynamic binding in a neural network for shape recognition. *Psychological Review*. In Press.

Kanade, T. (1981). Recovery of the three-dimensional shape of an object from a single view. *Artificial Intelligence*, *17*, 409–460.

Lowe, D. G. (1985). *Perceptual Organization and Visual Recognition*. Boston: Kluwer Academic Publishers.

Lowe, D. G., & Binford, T. O. (1982). Segmentation and aggregation: An approach to figure-ground phenomena. In *Proceedings of the DARPA IU Workshop* (pp. 168–178). Palo Alto, CA: (null).

Marr, D. (1982). *Vision*. San Francisco: Freeman.

Rock, I., & Palmer, S. E. (1990). The legacy of Gestalt psychology. *Scientific American*, *263*, 84–90.

von der Malsburg, C. (1981). *The correlation theory of brain function* (Internal Report 81-2). Goettingen: Department of Neurobiology, Max Planck Intitute for Biophysical Chemistry.

von der Malsburg, C., & Schneider, W. (1986). A neural cocktail-party processor. *Biological Cybernetics*, *54*, 29–40.